# Sample complexity of policy search with known dynamics

**Peter L. Bartlett**
Divison of Computer Science and Department of Statistics
University of California, Berkeley
Berkeley, CA 94720-1776
bartlett@cs.berkeley.edu

**Ambuj Tewari**
Division of Computer Science
University of California, Berkeley
Berkeley, CA 94720-1776
ambuj@cs.berkeley.edu

## Abstract

We consider methods that try to find a good policy for a Markov decision process by choosing one from a given class. The policy is chosen based on its empirical performance in simulations. We are interested in conditions on the complexity of the policy class that ensure the success of such simulation based policy search methods. We show that under bounds on the amount of computation involved in computing policies, transition dynamics and rewards, uniform convergence of empirical estimates to true value functions occurs. Previously, such results were derived by assuming boundedness of pseudodimension and Lipschitz continuity. These assumptions and ours are both stronger than the usual combinatorial complexity measures. We show, via minimax inequalities, that this is essential: boundedness of pseudodimension or fat-shattering dimension alone is not sufficient.

## 1 Introduction

A Markov Decision Process (MDP) models a situation in which an agent interacts (by performing actions and receiving rewards) with an environment whose dynamics is Markovian, i.e. the future is independent of the past given the current state of the environment. Except for toy problems with a few states, computing an optimal policy for an MDP is usually out of the question. Some relaxations need to be done if our aim is to develop tractable methods for achieving near optimal performance. One possibility is to avoid considering all possible policies by restricting oneself to a smaller class $\Pi$ of policies. Given a simulator for the environment, we try to pick the best policy from $\Pi$. The hope is that if the policy class is appropriately chosen, the best policy in $\Pi$ would not be too much worse than the true optimal policy.

Use of simulators introduces an additional issue: how is one to be sure that performance of policies in the class $\Pi$ on a few simulations is indicative of their true performance? This is reminiscent of the situation in statistical learning. There the aim is to learn a concept and one restricts attention to a hypotheses class which may or may not contain the "true" concept. The sample complexity question then is: how many labeled examples are needed in order to be confident that error rates on the training set are close to the true error rates of the hypotheses in our class? The answer turns out to depend on "complexity" of the hypothesis class as measured by combinatorial quantities associated with the class such as the VC dimension, the pseudodimension and the fat-shattering dimension.

Some progress [6,7] has already been made to obtain uniform bounds on the difference between value functions and their empirical estimates, where the value function of a policy is the expected long term reward starting from a certain state and following the policy thereafter. We continue this line of work by further investigating what properties of the policy class determine the rate of uniform convergence of value function estimates. The key difference between the usual statistical learning setting and ours is that we not only have to consider the complexity of the class $\Pi$ but also of the

classes derived from $\Pi$ by composing the functions in $\Pi$ with themselves and with the state evolution process implied by the simulator.

Ng and Jordan [7] used a finite pseudodimension condition along with Lipschitz continuity to derive uniform bounds. The Lipschitz condition was used to control the covering numbers of the iterated function classes. We provide a uniform convergence result (Theorem 1) under the assumption that policies are parameterized by a finite number of parameters and that the computations involved in computing the policy, the single-step simulation function and the reward function all require a bounded number of arithmetic operations on real numbers. The number of samples required grows linearly with the dimension of the parameter space but is independent of the dimension of the state space. Ng and Jordan's and our assumptions are both stronger than just assuming finiteness of some combinatorial dimension. We show that this is unavoidable by constructing two examples where the fat-shattering dimension and the pseudodimension respectively are bounded, yet no simulation based method succeeds in estimating the true values of policies well. This happens because iteratively composing a function class with itself can quickly destroy finiteness of combinatorial dimensions. Additional assumptions are therefore needed to ensure that these iterates continue to have bounded combinatorial dimensions.

Although we restrict ourselves to MDPs for ease of exposition, the analysis in this paper carries over easily to the case of partially obervable MDPs (POMDPs), provided the simulator also simulates the conditional distribution of observations given state using a bounded amount of computation. The plan of the rest of the paper is as follows. We set up notation and terminology in Section 2. In the same section, we describe the model of computation over reals that we use. Section 3 proves Theorem 1, which gives a sample complexity bound for achieving a desired level of performance within the policy class. In Section 4, we give two examples of policy classes whose combinatorial dimensions are bounded. Nevertheless, we can prove strong minimax lower bounds implying that no method of choosing a policy based on empirical estimates can do well for these examples.

## 2   Preliminaries

We define an MDP $\mathcal{M}$ as a tuple $(S, D, A, P(\cdot|s, a), r, \gamma)$ where $S$ is the state space, $D$ the initial state distribution, $A$ the action space, $P(s'|s, a)$ gives the probability of moving to state $s'$ upon taking action $a$ in state $s$, $r$ is a function mapping states to distributions over rewards (which are assumed to lie in a bounded interval $[0, R]$), and $\gamma \in (0, 1)$ is a factor that discounts future rewards. In this paper, we assume that the state space $S$ and the action space $A$ are finite dimensional Euclidean spaces of dimensionality $d_S$ and $d_A$ respectively.

A (randomized) policy $\pi$ is a mapping from $S$ to distributions over $A$. Each policy $\pi$ induces a natural Markov chain on the state space of the MDP, namely the one obtained by starting in a start state $s_0$ sampled from $D$ and $s_{t+1}$ sampled according to $P(\cdot|s_t, a_t)$ with $a_t$ drawn from $\pi(s_t)$ for $t \geq 0$. Let $r_t(\pi)$ be the expected reward at time step $t$ in this Markov chain, i.e. $r_t(\pi) = \mathbb{E}[\rho_t]$ where $\rho_t$ is drawn from the distribution $r(s_t)$. Note that the expectation is over the randomness in the choice of the initial state, the state transitions, and the randomized policy and reward outcomes. Define the value $V_{\mathcal{M}}(\pi)$ of the policy by

$$V_{\mathcal{M}}(\pi) = \sum_{t=0}^{\infty} \gamma^t r_t(\pi) \ .$$

We omit the subscript $\mathcal{M}$ in the value function if the MDP in question is unambiguously identified. For a class $\Pi$ of policies, define

$$\text{opt}(\mathcal{M}, \Pi) = \sup_{\pi \in \Pi} V_{\mathcal{M}}(\pi) \ .$$

The *regret* of a policy $\pi'$ relative to an MDP $\mathcal{M}$ and a policy class $\Pi$ is defined as

$$\text{Reg}_{\mathcal{M}, \Pi}(\pi') = \text{opt}(\mathcal{M}, \Pi) - V_{\mathcal{M}}(\pi') \ .$$

We use a degree bounded version of the Blum-Shub-Smale [3] model of computation over reals. At each time step, we can perform one of the four arithmetic operations $+, -, \times, /$ or can branch based on a comparison (say $<$). While Blum et al. allow an arbitrary fixed rational map to be computed in one time step, we further require that the degree of any of the polynomials appearing at computation nodes be at most 1.

**Definition 1.** *Let $k, l, m, \tau$ be positive integers, $f$ a function from $\mathbb{R}^k$ to probability distributions over $\mathbb{R}^l$ and $\Xi$ a probability distribution over $\mathbb{R}^m$. The function $f$ is $(\Xi, \tau)$-**computable** if there exists a degree bounded finite dimensional machine $\mathbb{M}$ over $\mathbb{R}$ with input space $\mathbb{R}^{k+m}$ and output space $\mathbb{R}^l$ such that the following hold.*

1. *For every $x \in \mathbb{R}^k$ and $\xi \in \mathbb{R}^m$, the machine halts with halting time $T_{\mathbb{M}}(x, \xi) \leq \tau$.*

2. *For every $x \in \mathbb{R}^k$, if $\xi \in \mathbb{R}^m$ is distributed according to $\Xi$ the input-output map $\Phi_{\mathbb{M}}(x, \xi)$ is distributed as $f(x)$.*

Informally, the definition states that given access to an oracle which generates samples from $\Xi$, we can generate samples from $f(x)$ by doing a bounded amount of computation. For precise definitions of the input-output map and halting time, we refer the reader to [3, Chap. 2].

In Section 3, we assume that the policy class $\Pi$ is parameterized by a finite dimensional parameter $\theta \in \mathbb{R}^d$. In this setting $\pi(s; \theta)$, $P(\cdot|s, a)$ and $r(s)$ are distributions over $\mathbb{R}^{d_A}$, $\mathbb{R}^{d_S}$ and $[0, R]$ respectively. The following assumption states that all these maps are computable within $\tau$ time steps in our model of computation.

**Assumption A.** *There exists a probability distribution $\Xi$ over $\mathbb{R}^m$ and a positive integer $\tau$ such that $\pi(s; \theta)$, $P(\cdot|s, a)$ and $r(s)$ are $(\Xi, \tau)$-computable. Let $\mathbb{M}_\pi$, $\mathbb{M}_P$ and $\mathbb{M}_r$ respectively be the machines that compute them.*

This assumption will be satisfied if we have three "programs" that make a call to a random number generator for distribution $\Xi$, do a fixed number of floating-point operations and simulate the policies in our class, the state-transition dynamics and the rewards respectively. The following two examples illustrate this for the state-transition dynamics.

- *Linear Dynamical System with Additive Noise* [1]
  Suppose $P$ and $Q$ are $d_S \times d_S$ and $d_S \times d_A$ matrices and the system dynamics is given by
  $$s_{t+1} = Ps_t + Qa_t + \xi_t \,, \tag{1}$$
  where $\xi_t$ are i.i.d. from some distribution $\Xi$. Since computing (1) takes $2(d_S^2 + d_S d_A + d_S)$ operations, $P(\cdot|s, a)$ is $(\Xi, \tau)$-computable for $\tau = O(d_S(d_S + d_A))$.

- *Discrete States and Actions*
  Suppose $S = \{1, 2, \ldots, n_S\}$ and $A = \{1, 2, \ldots, n_A\}$. For some fixed $s, a$, $P(\cdot|s, a)$ is described by $n$ numbers $\vec{p}_{s,a} = (p_1, \ldots, p_{n_S})$, $\sum_i p_i = 1$. Let $P_k = \sum_{i=1}^{k} p_i$. For $\xi \in (0, 1]$, set $f(\xi) = \min\{k : P_k \geq \xi\}$. Thus, if $\xi$ has uniform distribution on $(0, 1]$, then $f(\xi) = k$ with probability $p_k$. Since the $P_k$'s are non-decreasing, $f(\xi)$ can be computed in $\log n_S$ steps using binary search. But this was for a fixed $s, a$ pair. Finding which $\vec{p}_{s,a}$ to use, further takes $\log(n_S n_A)$ steps using binary search. So if $\Xi$ denotes the uniform distribution on $(0, 1]$ then $P(\cdot|s, a)$ is $(\Xi, \tau)$-computable for $\tau = O(\log n_S + \log n_A)$.

For a small $\epsilon$, let $H$ be the $\epsilon$ horizon time, i.e. ignoring rewards beyond time $H$ does not affect the value of any policy by more than $\epsilon$. To obtain sample rewards, given initial state $s_0$ and policy $\pi_\theta = \pi(\cdot; \theta)$, we first compute the trajectory $s_0, \ldots, s_H$ sampled from the Markov chain induced by $\pi_\theta$. This requires $H$ "calls" each to $\mathbb{M}_\pi$ and $\mathbb{M}_P$. A further $H + 1$ calls to $\mathbb{M}_r$ are then required to generate the rewards $\rho_0$ through $\rho_H$. These calls require a total of $3H + 1$ samples from $\Xi$. The empirical estimates are computed as follows. Suppose, for $1 \leq i \leq n$, $(s_0^{(i)}, \vec{\xi}_i)$ are i.i.d. samples generated from the joint distribution $D \times \Xi^{3H+1}$. Define the empirical estimate of the value of the policy $\pi$ by

$$\hat{V}_{\mathcal{M}}^H(\pi_\theta) = \frac{1}{n} \sum_{i=1}^{n} \sum_{t=0}^{H} \gamma^t \rho_t(s_0^{(i)}, \theta, \vec{\xi}_i) \,.$$

We omit the subscript $\mathcal{M}$ in $\hat{V}$ when it is clear from the context. Define an $\epsilon$-approximate maximizer of $\hat{V}$ to be a policy $\pi'$ such that

$$\hat{V}_{\mathcal{M}}^H(\pi') \geq \sup_{\pi \in \Pi} \hat{V}_{\mathcal{M}}^H(\pi) - \epsilon \,.$$

Finally, we mention the definitions of three standard combinatorial dimensions. Let $\mathcal{X}$ be some space and consider classes $\mathcal{G}$ and $\mathcal{F}$ of $\{-1, +1\}$ and real valued functions on $\mathcal{X}$, respectively. Fix a finite set $X = \{x_1, \ldots, x_n\} \subseteq \mathcal{X}$. We say that $\mathcal{G}$ *shatters* $X$ if for all bit vectors $\vec{b} \in \{0, 1\}^n$ there exists $g \in \mathcal{G}$ such that for all $i$, $b_i = 0 \Rightarrow g(x_i) = -1$, $b_i = 1 \Rightarrow g(x_i) = +1$. We say that $\mathcal{F}$ *shatters* $X$ if there exists $\vec{r} \in \mathbb{R}^n$ such that, for all bit vectors $\vec{b} \in \{0, 1\}^n$, there exists $f \in \mathcal{F}$ such that for all $i$, $b_i = 0 \Rightarrow f(x_i) < r_i$, $b_i = 1 \Rightarrow f(x_i) \geq r_i$. We say that $\mathcal{F}$ $\epsilon$-*shatters* $X$ if these exists $\vec{r} \in \mathbb{R}^n$ such that, for all bit vectors $\vec{b} \in \{0, 1\}^n$, there exists $f \in \mathcal{F}$ such that for all $i$, $b_i = 0 \Rightarrow f(x_i) \leq r_i - \epsilon$, $b_i = 1 \Rightarrow f(x_i) \geq r_i + \epsilon$. We then have the following definitions,

$$\text{VCdim}(\mathcal{G}) = \max\{|X| : \mathcal{G} \text{ shatters } X\},$$
$$\text{Pdim}(\mathcal{F}) = \max\{|X| : \mathcal{F} \text{ shatters } X\},$$
$$\text{fat}_{\mathcal{F}}(\epsilon) = \max\{|X| : \mathcal{F} \ \epsilon\text{-shatters } X\}.$$

# 3 Regret Bound for Parametric Policy Classes Computable in Bounded Time

**Theorem 1.** *Fix an MDP $\mathcal{M}$, a policy class $\Pi = \{s \mapsto \pi(s; \theta) : \theta \in \mathbb{R}^d\}$, and an $\epsilon > 0$. Suppose Assumption A holds. Then*

$$n > O\left(\frac{R^2 H d\tau}{(1-\gamma)^2 \epsilon^2} \log \frac{R}{\epsilon(1-\gamma)}\right)$$

*ensures that $\mathbb{E}\left[\text{Reg}_{\mathcal{M},\Pi}(\pi_n)\right] \leq 3\epsilon + \epsilon'$, where $\pi_n$ is an $\epsilon'$-approximate maximizer of $\hat{V}$ and $H = \log_{1/\gamma}(2R/(\epsilon(1-\gamma)))$ is the $\epsilon/2$ horizon time.*

*Proof.* The proof consists of three steps: (1) Assumption A is used to get bounds on pseudodimension; (2) The pseudodimension bound is used to prove uniform convergence of empirical estimates to true value functions; (3) Uniform convergence and the definition of $\epsilon'$-approximate maximizer gives the bound on expected regret.

STEP 1. Given initial state $s_0$, parameter $\theta$ and random numbers $\xi_1$ through $\xi_{3H+1}$, we first compute the trajectory as follows. Recall that $\Phi_{\mathbb{M}}$ refers to the input-output map of a machine $\mathbb{M}$.

$$s_t = \Phi_{\mathbb{M}_P}(s_{t-1}, \Phi_{\mathbb{M}_\pi}(\theta, s, \xi_{2t-1}), \xi_{2t}), \ 1 \leq t \leq H. \tag{2}$$

The rewards are then computed by

$$\rho_t = \Phi_{\mathbb{M}_r}(s_t, \xi_{2H+t+1}), \ 0 \leq t \leq H. \tag{3}$$

The $H$-step discounted reward sum is computed as

$$\sum_{t=0}^{H} \gamma^t \rho_t = \rho_0 + \gamma(\rho_1 + \gamma(\rho_2 + \ldots (p_{H-1} + \gamma\rho_H)\ldots)). \tag{4}$$

Define the function class $\mathcal{R} = \{(s_0, \vec{\xi}) \mapsto \sum_{t=0}^{H} \gamma^t \rho_t(s_0, \theta, \vec{\xi}) : \theta \in \mathbb{R}^d\}$, where we have explicitly shown the dependence of $\rho_t$ on $s_0$, $\theta$ and $\vec{\xi}$. Let us count the number of arithmetic operations needed to compute a function in this class. Using Assumption A, we see that steps (2) and (3) require no more than $2\tau H$ and $\tau(H + 1)$ operations respectively. Step (4) requires $H$ multiplications and $H$ additions. This gives a total of $2\tau H + \tau(H + 1) + 2H \leq 6\tau H$ operations. Goldberg and Jerrum [4] showed that the VC dimension of a function class can be bounded in terms of an upper bound on the number of arithmetic operations it takes to compute the functions in the class. Since the pseudodimension of $\mathcal{R}$ can be written as

$$\text{Pdim}(\mathcal{R}) = \text{VCdim}\{(s_0, \vec{\xi}, c) \mapsto \text{sign}(f(s_0, \vec{\xi}) - c) : f \in \mathcal{R}, c \in \mathbb{R}\},$$

we get the following bound by [2, Thm. 8.4],

$$\text{Pdim}(\mathcal{R}) \leq 4d(6\tau H + 3). \tag{5}$$

STEP 2. Let $V^H(\pi) = \sum_{t=0}^{H} \gamma^t r_t(\pi)$. For the choice of $H$ stated in the theorem, we have for all $\pi$, $|V^H(\pi) - V(\pi)| \leq \epsilon/2$. Therefore,

$$P^n(\exists \pi \in \Pi : |\hat{V}^H(\pi) - V(\pi)| > \epsilon) \leq P^n(\exists \pi \in \Pi : |\hat{V}^H(\pi) - V^H(\pi)| > \epsilon/2). \tag{6}$$

Functions in $\mathcal{R}$ are positive and bounded above by $R' = R/(1-\gamma)$. There are well-known bounds for deviations of empirical estimates from true expectations for bounded function classes in terms of the pseudodimension of the class (see, for example, Theorems 3 and 5 in [5]; also see Pollard's book [8]). Using a weak form of these results, we get

$$P^n(\exists \pi \in \Pi : |\hat{V}^H(\pi) - V^H(\pi)| > \epsilon) \le 8 \left( \frac{32eR'}{\epsilon} \right)^{2\,\mathrm{Pdim}(\mathcal{R})} e^{-\epsilon^2 n/64R'^2} \ .$$

In order to ensure that $P^n(\exists \pi \in \Pi : |\hat{V}^H(\pi) - V^H(\pi)| > \epsilon/2) < \delta$, we need

$$8 \left( \frac{64eR'}{\epsilon} \right)^{2\,\mathrm{Pdim}(\mathcal{R})} e^{-\epsilon^2 n/256R'^2} < \delta \ ,$$

Using the bound (5) on $\mathrm{Pdim}(\mathcal{R})$, we get that

$$P^n \left( \sup_{\pi \in \Pi} \left| \hat{V}^H(\pi) - V(\pi) \right| > \epsilon \right) < \delta \ , \tag{7}$$

provided

$$n > \frac{256R^2}{(1-\gamma)^2 \epsilon^2} \left( \log \left( \frac{8}{\delta} \right) + 8d(6\tau H + 3) \log \left( \frac{64eR}{(1-\gamma)\epsilon} \right) \right) \ .$$

STEP 3. We now show that (7) implies $\mathbb{E}\,\mathrm{Reg}_{\mathcal{M},\Pi}(\pi_n) \le R\delta/(1-\gamma) + (2\epsilon + \epsilon')$. The theorem them immediately follows by setting $\delta = (1-\gamma)\epsilon/R$.

Suppose that for all $\pi \in \Pi$, $|\hat{V}^H(\pi) - V(\pi)| \le \epsilon$. This implies that for all $\pi \in \Pi$, $V(\pi) \le \hat{V}^H(\pi) + \epsilon$. Since $\pi_n$ is an $\epsilon'$-approximate maximizer of $\hat{V}$, we have for all $\pi \in \Pi$, $\hat{V}^H(\pi) \le \hat{V}^H(\pi_n) + \epsilon'$. Thus, for all $\pi \in \Pi$, $V(\pi) \le \hat{V}^H(\pi_n) + \epsilon + \epsilon'$. Taking the supremum over $\pi \in \Pi$ and using the fact that $\hat{V}^H(\pi_n) \le V(\pi_n) + \epsilon$, we get $\sup_{\pi \in \Pi} V(\pi) \le V(\pi_n) + 2\epsilon + \epsilon'$, which is equivalent to $\mathrm{Reg}_{\mathcal{M},\Pi}(\pi_n) \le 2\epsilon + \epsilon'$. Thus, if (7) holds then we have

$$P^n \left( \mathrm{Reg}_{\mathcal{M},\Pi}(\pi_n) > 2\epsilon + \epsilon' \right) < \delta \ .$$

Denoting the event $\{ \mathrm{Reg}_{\mathcal{M},\Pi}(\pi_n) > 2\epsilon + \epsilon' \}$ by $E$, we have

$$\mathbb{E}\,\mathrm{Reg}_{\mathcal{M},\Pi}(\pi_n) = \mathbb{E}\,\mathrm{Reg}_{\mathcal{M},\Pi}(\pi_n)1_E + \mathbb{E}\,\mathrm{Reg}_{\mathcal{M},\Pi}(\pi_n)1_{(\neg E)}$$
$$\le R\delta/(1-\gamma) + (2\epsilon + \epsilon') \ .$$

where we used the fact that regret is bounded above by $R/(1-\gamma)$. $\qquad\square$

## 4  Two Policy Classes Having Bounded Combinatorial Dimensions

We will describe two policy classes for which we can prove that there are strong limitations on the performance of any method (of choosing a policy out of a policy class) that has access only to empirically observed rewards. Somewhat surprisingly, one can show this for policy classes which are "simple" in the sense that standard combinatorial dimensions of these classes are bounded. This shows that sufficient conditions for the success of simulation based policy search (such as the assumptions in [7] and in our Theorem 1) have to be necessarily stronger than boundedness of standard combinatorial dimensions.

The first example is a policy class $\mathcal{F}_1$ for which $\mathrm{fat}_{\mathcal{F}_1}(\epsilon) < \infty$ for all $\epsilon > 0$. The second example is a class $\mathcal{F}_2$ for which $\mathrm{Pdim}(\mathcal{F}_2) = 1$. Since finiteness of pseudodimension is a stronger condition, the second example makes our point more forcefully than the first one. However, the first example is considerably less contrived than the second one.

**Example 1**

Let $\mathcal{M}_D = (S, D, A, P(\cdot|s,a), r, \gamma)$ be an MDP where $S = [-1, +1]$, $D = $ some distribution on $[-1, +1]$, $A = [-2, +2]$,

$$P(s'|s,a) = 1 \text{ if } s' = \max(-1, \min(s + a, 1))), \ 0 \text{ otherwise} \ ,$$

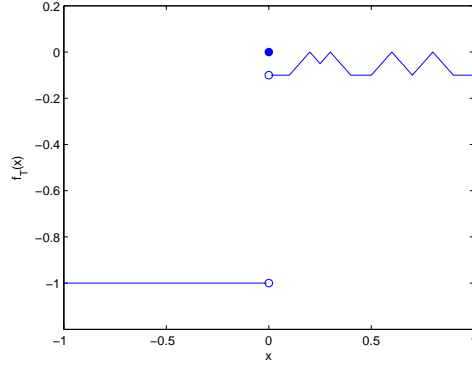

Figure 1: Plot of the function $f_T$ with $T = \{0.2, 0.3, 0.6, 0.8\}$. Note that, for $x > 0$, $f_T(x)$ is 0 iff $x \in T$. Also, $f_T(x)$ satisfies the Lipschitz condition (with constant 1) everywhere except at 0.

$r =$ deterministic reward that maps $s$ to $s$, and $\gamma =$ some fixed discount factor in $(0, 1)$.

For a function $f : [-1, +1] \mapsto [-1, +1]$, let $\pi_f$ denote the (deterministic) policy which takes action $f(s) - s$ in state $s$. Given a class $\mathcal{F}$ of functions, we define an associated policy class $\Pi_{\mathcal{F}} = \{\pi_f : f \in \mathcal{F}\}$.

We now describe a specific function class $\mathcal{F}_1$. Fix $\epsilon_1 > 0$. Let $T$ be an arbitrary finite subset of $(0, 1)$. Let $\delta(x) = (1 - |x|)_+$ be the "triangular spike" function. Let

$$
f_T(x) = \begin{cases} -1 & -1 \le x < 0 \\ 0 & x = 0 \\ \max_{y \in T} \left( \frac{\epsilon_1}{|T|} \delta \left( \frac{x-y}{\epsilon_1/|T|} \right) - \frac{\epsilon_1}{|T|} \right) & 0 < x \le 1 \end{cases}.
$$

There is a spike at each point in $T$ and the tips of the spikes just touch the $X$-axis (see Figure 1). Since $-1$ and 0 are fixed points of $F_T(x)$, it is straightforward to verify that

$$
f_T^2(x) = \begin{cases} -1 & -1 \le x < 0 \\ 0 & x = 0 \\ 1_{(x \in T)} - 1 & 0 < x \le 1 \end{cases}. \tag{8}
$$

Also, $f_T^n = f_T^2$ for all $n > 2$. Define $\mathcal{F}_1 = \{f_T : T \subset (\epsilon_1, 1), |T| < \infty\}$. By construction, functions in $\mathcal{F}_1$ have bounded total variation and so, $\mathrm{fat}_{\mathcal{F}_1}(\epsilon)$ is $O(1/\epsilon)$ (see, for example, [2, Chap. 11]). Moreover, $f_T(x)$ satisfies the Lipschitz condition everywhere (with constant $L = 1$) except at 0. This is striking in the sense that the loss of the Lipschitz property at a single point allows us to prove the following lower bound.

**Theorem 2.** *Let $g_n$ range over functions from $S^n$ to $\mathcal{F}_1$. Let $D$ range over probability distributions on $S$. Then,*

$$
\inf_{g_n} \sup_D \mathbb{E}_{(s^1, \ldots, s^n) \sim D^n} \left[ \mathrm{Reg}_{\mathcal{M}_D, \Pi_{\mathcal{F}_1}} (\pi_{g_n(s^1, \ldots, s^n)}) \right] \ge \frac{\gamma^2}{1 - \gamma} - 2\epsilon_1 .
$$

This says that for any method that maps random initial states $s^1, \ldots, s^n$ to a policy in $\Pi_{\mathcal{F}_1}$, there is an initial state distribution such that the expected regret of the selected policy is at least $\gamma^2/(1 - \gamma) - 2\epsilon_1$. This is in sharp contrast to Theorem 1 where we could reduce, by using sufficiently many samples, the expected regret down to any positive number given the ability to maximize the empirical estimates $\hat{V}$.

Let us see how maximization of empirical estimates behaves in this case. Since $\mathrm{fat}_{\mathcal{F}_1}(\epsilon) < \infty$ for all $\epsilon > 0$, the law of large numbers holds uniformly [1, Thm. 2.5] over the class $\mathcal{F}_1$. The transitions, policies and rewards here are all deterministic. The reward function is just the identity. This means that the 1-step reward function family is just $\mathcal{F}_1$. So the estimates of 1-step rewards are

still uniformly concentrated around their expected values. Since the contribution of rewards from time step 2 onwards can be no more than $\gamma^2 + \gamma^3 + \ldots = \gamma^2/(1-\gamma)$, we can claim that the expected regret of the $\hat{V}$ maximizer $\pi_n$ behaves like

$$\mathbb{E}\left[\mathrm{Reg}_{\mathcal{M},\Pi_{\mathcal{F}_1}}(\pi_n)\right] \leq \frac{\gamma^2}{1-\gamma} + e_n$$

where $e_n \to 0$. Thus the bound in Theorem 2 above is essentially tight.

Before we prove Theorem 2, we need the following lemma whose proof is given in the appendix accompanying the paper.

**Lemma 1.** *Fix an interval $(a, b)$ and let $\mathcal{T}$ be the set of all its finite subsets. Let $g_n$ range over functions from $(a, b)^n$ to $\mathcal{T}$. Let $D$ range over probability distributions on $(a, b)$. Then,*

$$\inf_{g_n} \sup_{D} \left( \sup_{T \in \mathcal{T}} \mathbb{E}_{X \sim D} 1_{(X \in T)} - \mathbb{E}_{(X_1, \ldots, X_n) \sim D^n} \mathbb{E}_{(X \sim D)} 1_{(X \in g_n(X_1, \ldots, X_n))} \right) \geq 1 \,.$$

*Proof of Theorem 2.* We will prove the inequality when $D$ ranges over distributions on $(0, 1)$ which, obviously, implies the theorem.

Since, for all $f \in \mathcal{F}_1$ and $n > 2$, $f^n = f^2$, we have

$$\mathrm{opt}(\mathcal{M}_D, \Pi_{\mathcal{F}_1}) - \mathbb{E}_{(s^1, \ldots, s^n) \sim D^n} V_{\mathcal{M}_D}(\pi_{g_n(s^1, \ldots, s^n)})$$

$$= \sup_{f \in \mathcal{F}_1} \mathbb{E}_{s \sim D} \left[ s + \gamma f(s) + \frac{\gamma^2}{1-\gamma} f^2(s) \right]$$

$$- \mathbb{E}_{(s^1, \ldots, s^n) \sim D^n} \left[ \mathbb{E}_{s \sim D}[s + \gamma g_n(s^1, \ldots, s^n)(s) + \frac{\gamma^2}{1-\gamma} g_n(s^1, \ldots, s^n)^2(s)] \right]$$

$$= \sup_{f \in \mathcal{F}_1} \mathbb{E}_{s \sim D} \left[ \gamma f(s) + \frac{\gamma^2}{1-\gamma} f^2(s) \right]$$

$$- \mathbb{E}_{(s^1, \ldots, s^n) \sim D^n} \left[ \mathbb{E}_{s \sim D}[\gamma g_n(s^1, \ldots, s^n)(s) + \frac{\gamma^2}{1-\gamma} g_n(s^1, \ldots, s^n)^2(s)] \right]$$

For all $f_1, f_2$, $|\mathbb{E}f_1 - \mathbb{E}f_2| \leq \mathbb{E}|f_1 - f_2| \leq \epsilon_1$. Therefore, we can get rid of the first terms in both sub-expressions above without changing the value by more than $2\gamma\epsilon_1$.

$$\geq \sup_{f \in \mathcal{F}_1} \mathbb{E}_{s \sim D} \left[ \frac{\gamma^2}{1-\gamma} f^2(s) \right] - \mathbb{E}_{(s^1, \ldots, s^n) \sim D^n} \left[ \mathbb{E}_{s \sim D}[\frac{\gamma^2}{1-\gamma} g_n(s^1, \ldots, s^n)^2(s)] \right]$$

$$- 2\gamma\epsilon_1$$

$$= \frac{\gamma^2}{1-\gamma} \left( \sup_{f \in \mathcal{F}_1} \mathbb{E}_{s \sim D} \left[ f^2(s) + 1 \right] - \mathbb{E}_{(s^1, \ldots, s^n) \sim D^n} \mathbb{E}_{s \sim D}[g_n(s^1, \ldots, s^n)^2(s) + 1] \right)$$

$$- 2\gamma\epsilon_1$$

From (8), we know that $f_T^2(x) + 1$ restricted to $x \in (0, 1)$ is the same as $1_{(x \in T)}$. Therefore, restricting $D$ to probability measures on $(0, 1)$ and applying Lemma 1, we get

$$\inf_{g_n} \sup_{D} \left( \mathrm{opt}(\mathcal{M}_D, \Pi_{\mathcal{F}_1}) - \mathbb{E}_{(s^1, \ldots, s^n) \sim D^n} V_{\mathcal{M}_D}(\pi_{g_n(s^1, \ldots, s^n)}) \right) \geq \frac{\gamma^2}{1-\gamma} - 2\gamma\epsilon_1 \,.$$

To finish the proof, we note that $\gamma < 1$ and, by definition,

$$\mathrm{Reg}_{\mathcal{M}_D, \Pi_{\mathcal{F}_1}}(\pi_{g_n(s^1, \ldots, s^n)}) = \mathrm{opt}(\mathcal{M}_D, \Pi_{\mathcal{F}_1}) - V_{\mathcal{M}_D}(\pi_{g_n(s^1, \ldots, s^n)}) \,.$$

$\square$

### Example 2

We use the MDP of the previous section with a different policy class which we now describe. For a real number $x, y \in (0, 1)$ with binary expansions (choose the terminating representation for rationals) $0.b_1 b_2 b_3 \ldots$ and $0.c_1 c_2 c_3 \ldots$, define

$$\mathrm{mix}(x, y) = 0.b_1 c_1 b_2 c_2 \ldots \qquad \mathrm{stretch}(x) = 0.b_1 0 b_2 0 b_3 \ldots$$
$$\mathrm{even}(x) = 0.b_2 b_4 b_6 \ldots \qquad \mathrm{odd}(x) = 0.b_1 b_3 b_5 \ldots$$

Some obvious identities are $\text{mix}(x, y) = \text{stretch}(x) + \text{stretch}(y)/2$, $\text{odd}(\text{mix}(x, y)) = x$ and $\text{even}(\text{mix}(x, y)) = y$. Now fix $\epsilon_2 > 0$. Since, finite subsets of $(0, 1)$ and irrationals in $(0, \epsilon_2)$ have the same cardinality, there exists a bijection $h$ which maps every finite subset $T$ of $(0, 1)$ to some irrational $h(T) \in (0, \epsilon_2)$. For a finite subset $T$ of $(0, 1)$, define

$$f_T(x) = \begin{cases} 0 & x = -1 \\ 1_{(\text{odd}(-x) \in h^{-1}(\text{even}(-x)))} & -1 < x < 0 \\ 0 & x = 0 \\ -\text{mix}(x, h(T)) & 0 < x < 1 \\ 1 & x = 1 \end{cases}.$$

It is easy to check that with this definition, $f_T^2(x) = 1_{(x \in T)}$ for $x \in (0, 1)$. Finally, let $\mathcal{F}_2 = \{f_T : T \subset (0, 1), |T| < \infty\}$. To calculate the pseudodimension of this class, note that using the identity $\text{mix}(x, y) = \text{stretch}(x) + \text{stretch}(y)/2$, every function $f_T$ in the class can be written as $f_T = f_0 + \tilde{f}_T$ where $f_0$ is a fixed function (does not depend on $T$) and $\tilde{f}_T$ is given by

$$\tilde{f}_T(x) = \begin{cases} 0 & -1 \le x \le 0 \\ -\text{stretch}(h(T))/2 & 0 < x < 1 \\ 0 & x = 1 \end{cases}.$$

Let $\mathcal{H} = \{\tilde{f}_T : T \subset (0, 1), |T| < \infty\}$. Since $\text{Pdim}(\mathcal{H} + f_0) = \text{Pdim}(\mathcal{H})$ for any class $\mathcal{H}$ and a fixed function $f_0$, we have $\text{Pdim}(\mathcal{F}_2) = \text{Pdim}(\mathcal{H})$. As each function $\tilde{f}_T(x)$ is constant on $(0, 1)$ and zero elsewhere, we cannot shatter even two points using $\mathcal{H}$. Thus, $\text{Pdim}(\mathcal{H}) = 1$.

**Theorem 3.** *Let $g_n$ range over functions from $S^n$ to $\mathcal{F}_2$. Let $D$ range over probability distributions on $S$. Then,*

$$\inf_{g_n} \sup_D \mathbb{E}_{(s^1, \ldots, s^n) \sim D^n} \left[ \text{Reg}_{\mathcal{M}_D, \Pi_{\mathcal{F}_1}} (\pi_{g_n(s^1, \ldots, s^n)}) \right] \ge \frac{\gamma^2}{1 - \gamma} - \epsilon_2 .$$

*Sketch.* Let us only check that the properties of $\mathcal{F}_1$ that allowed us to proceed with the proof of Theorem 2 are also satisfied by $\mathcal{F}_2$. First, for all $f \in \mathcal{F}_2$ and $n > 2$, $f^n = f^2$. Second, for all $f_1, f_2 \in \mathcal{F}_2$ and $x \in [-1, +1]$, $|f_1(x) - f_2(x)| \le \epsilon_2/2$. This is because $f_{T_1}$ and $f_{T_2}$ can differ only for $x \in (0, 1)$. For such an $x$, $|f_{T_1}(x) - f_{T_2}(x)| = |\text{mix}(x, h(T_1)) - \text{mix}(x, h(T_2))| = |\text{stretch}(h(T_1)) - \text{stretch}(h(T_2))|/2 \le \epsilon_2/2$. Third, the restriction of $f_T^2$ to $(0, 1)$ is $1_{(x \in T)}$. $\square$

## Acknowledgments

We acknowledge the support of DARPA under grants HR0011-04-1-0014 and FA8750-05-2-0249.

## Footnotes

[1] In this case, the realizable dynamics (mapping from state to next state for a given policy class) is not uniformly Lipschitz if policies allow unbounded actions. So previously known bounds [7] are not applicable even in this simple setting.

## References

[1] Alon, N., Ben-David, S., Cesa-Bianchi, N. & Haussler, D. (1997) Scale-sensitive Dimensions, Uniform Convergence, and Learnability. *Journal of the ACM* **44**(4):615–631.

[2] Anthony, M. & Bartlett P.L. (1999) *Neural Network Learning: Theoretical Foundations.* Cambridge University Press.

[3] Blum, L., Cucker, F., Shub, M. & Smale, S. (1998) *Complexity and Real Computation.* Springer-Verlag.

[4] Goldberg, P.W. & Jerrum, M.R. (1995) Bounding the Vapnik-Chervonenkis Dimension of Concept Classes Parameterized by Real Numbers. *Machine Learning* **18**(2-3):131–148.

[5] Haussler, D. (1992) Decision Theoretic Generalizations of the PAC Model for Neural Net and Other Learning Applications. *Information and Computation* **100**:78–150.

[6] Jain, R. & Varaiya, P. (2006) Simulation-based Uniform Value Function Estimates of Discounted and Average-reward MDPs. *SIAM Journal on Control and Optimization*, to appear.

[7] Ng A.Y. & Jordan M.I. (2000) PEGASUS: A Policy Search Method for MDPs and POMDPs. In *Proceedings of the 16th Annual Conference on Uncertainty in Artificial Intelligence*, pp. 405–415. Morgan Kauffman Publishers.

[8] Pollard D. (1990) *Empirical Processes: Theory and Applications.* NSF-CBMS Regional Conference Series in Probability and Statistics, Volume 2.
